# Computational Differences between Asymmetrical and Symmetrical Networks

**Zhaoping Li**       **Peter Dayan**

Gatsby Computational Neuroscience Unit

17 Queen Square, London, England, WC1N 3AR.

zhaoping@gatsby.ucl.ac.uk       dayan@gatsby.ucl.ac.uk

## Abstract

Symmetrically connected recurrent networks have recently been used as models of a host of neural computations. However, because of the separation between excitation and inhibition, biological neural networks are asymmetrical. We study characteristic differences between asymmetrical networks and their symmetrical counterparts, showing that they have dramatically different dynamical behavior and also how the differences can be exploited for computational ends. We illustrate our results in the case of a network that is a selective amplifier.

## 1  Introduction

A large class of non-linear recurrent networks, including those studied by Grossberg,[9] the Hopfield net,[10,11] and many more recent proposals for the head direction system,[27] orientation tuning in primary visual cortex,[25,1,3,18] eye position,[20] and spatial location in the hippocampus[19] make a key simplifying assumption that the connections between the neurons are symmetric. Analysis is relatively straightforward in this case, since there is a Lyapunov (or energy) function[4,11] that often guarantees the convergence of the motion trajectory to an equilibrium point. However, the assumption of symmetry is broadly false. Networks in the brain are almost never symmetrical, if for no other reason than the separation between excitation and inhibition. In fact, the question of whether ignoring the polarity of the cells is simplification or over-simplication has yet to be fully answered.

Networks with excitatory and inhibitory cells (EI systems, for short) have long been studied,[6] for instance from the perspective of pattern generation in invertebrates,[23] and oscillations in the thalamus[7,24] and the olfactory system.[17,13] Further, since the discovery of 40 Hz oscillations or synchronization amongst cells in primary visual cortex of anesthetised cat,[8,5] oscillatory models of V1 involving separate excitatory and inhibitory cells have also been popular, mainly from the perspective of how the oscillations can be created and sustained and how they can

be used for feature linking or binding.[26, 22, 12] However the scope for computing with dynamically stable behaviors such as limit cycles is not yet clear.

In this paper, we study the computational differences between a family of EI systems and their symmetric counterparts (which we call S systems). One inspiration for this work is Li's nonlinear EI system modeling how the primary visual cortex performs contour enhancement and pre-attentive region segmentation.[14, 15] Studies by Braun[2] had suggested that an S system model of the cortex can not perform contour enhancement unless additional (and biologically questionable) mechanisms are used. This posed a question about the true differences between EI and S systems that we answer. We show that EI systems can take advantage of dynamically stable modes that are not available to S systems. The computational significance of this result is discussed and demonstrated in the context of models of orientation selectivity. More details of this work, especially its significance for models of the primary visual cortical system, can be found in Li & Dayan (1999).[16]

## 2 Theory and Experiment

Consider a simple, but biologically significant, EI system in which excitatory and inhibitory cells come in pairs and there are no 'long-range' connections from the inhibitory cells[14, 15] (to which the Lyapunov theory[13, 21] does not yet apply):

$$\dot{x}_i = -x_i + \sum_j J_{ij}g(x_j) - h(y_i) + I_i \qquad \tau_y \dot{y}_i = -y_i + \sum_j W_{ij}g(x_j), \qquad (1)$$

where $x_i$ are the principal excitatory cells, which receive external or sensory input $I_i$, and generate the network outputs $g(x_i)$; $y_i$ are the inhibitory interneurons (which are taken here as having no external input); function $g(x) = [x - T]_+$ is the threshold non-linear activation function for the excitatory cells; $h(y)$ is the activation function for the inhibitory cells (for analytical convenience, we use the linear form $h(y) = (y - T_y)$ although the results are similar with the non-linear $h(y) = [y - T_y]_+$); $\tau_y$ is a time-constant for the inhibitory cells; and $J_{ij}$ and $W_{ij}$ are the output connections of the excitatory cells. Excitatory and inhibitory cells can also be perturbed by Gaussian noise.

In the limit that the inhibitory cells are made infinitely fast ($\tau_y = 0$), we have $y_i = \sum_j W_{ij}g(x_j)$, leaving the excitatory cells to interact directly with each other:

$$\dot{x}_i = -x_i + \sum_j J_{ij}g(x_j) - h(\sum_j W_{ij}g(x_j)) + I_i \qquad (2)$$

$$= -x_i + \sum_j (J_{ij} - W_{ij})g(x_j) + I_i + \kappa_i \qquad (3)$$

where $\kappa_i$ are constants. In this network, the neural connections $J_{ij} - W_{ij}$ between any two cells $x$ can be either excitatory or inhibitory, as in many abstract neural network models. When $J_{ij} = J_{ji}$ and $W_{ij} = W_{ji}$, the network has symmetric connections. This paper compares EI systems with such connections and the corresponding S systems. Since there are many ways of setting $J_{ij}$ and $W_{ij}$ in the EI system whilst keeping constant $J_{ij} - W_{ij}$, which is the effective weight in the S system, one may intuitively expect the EI system to have a broader computational range.

The response of either system to given inputs is governed by the location and linear stability of their fixed points. The S network is so defined as to have fixed points $\bar{x}$ (where $\dot{x} = 0$ in equation 3) that are the same as those $(\bar{x}, \bar{y})$ of the EI network. In particular, $\bar{x}$ depends on inputs $I$ (the input-output sensitivity) via $d\bar{x} = (\mathbb{I} - JD_g + WD_g)^{-1} dI$, where $\mathbb{I}$ is the identity matrix, $J$ and $W$ are the connection matrices, and $D_g$ is a diagonal matrix with elements $[D_g]_{ii} = g'(\bar{x}_i)$. However, although the locations of the fixed points are the same for the EI and S

systems, the dynamical behavior of the systems about those fixed points are quite different, and this is what leads to their differing computational power.

To analyse the stability of the fixed points, consider, for simplicity the case that $\tau_y = 1$ in the EI system, and that the matrices $\mathbf{JD}_g$ and $\mathbf{WD}_g$ commute with eigenvalues $\lambda_k^J$ and $\lambda_k^W$ respectively for $k = 1, \ldots, N$ where $N$ is the dimension of $\mathbf{x}$. The local deviations near the fixed points along each of the $N$ modes will grow in time if the real parts of the following values are positive

$$\gamma_k^{EI} = -1 + (1/2\lambda_k^J \pm (\tfrac{1}{4}\left(\lambda_k^J\right)^2 - \lambda_k^W)^{1/2} \quad \text{for the EI system}$$
$$\gamma_k^S = -1 - \lambda_k^W + \lambda_k^J \quad \text{for the S system}$$

In the case that $\lambda^J$ and $\lambda^W$ are real, then *if* the S system is unstable, *then* the EI system is also unstable. For if $-1 + \lambda_i^J - \lambda_i^W > 0$ then $(\lambda_i^J)^2 - 4\lambda_i^W > (\lambda_i^J - 2)^2$, and so $2\gamma_i^{EI} = -2 + \lambda_i^J + ((\lambda_i^J)^2 - 4\lambda^W)^{1/2} > 0$. However, if the EI system is oscillatory, $4\lambda^W > (\lambda^J)^2$, then the S system is stable since $-1 + \lambda^J - \lambda^W < -1 + \lambda^J - (\lambda^J)^2/4 = -(1 - \lambda^J/2)^2 \leq 0$. Hence the EI system can be unstable and oscillatory while the S system is stable.

We are interested in the capacity of both systems to be selective *amplifiers*. This means that there is a class of inputs $\mathbf{I}$ that should be comparatively boosted by the system; whereas others should be comparatively suppressed. For instance, if the cells represent the orientation of a bar at a point, then the mode containing a unimodal, well-tuned, 'bump' in orientation space should be enhanced compared with poorly tuned inputs.[25,1,18] However, if the cells represent oriented small bars at multiple points in visual space, then isolated smooth and straight contours should be enhanced compared with extended homogeneous textures.[14,15]

The quality of the systems will be judged according to how much selective amplification they can stably deliver. The critical trade-off is that the more the selected mode is amplified, the more likely it is that, when the input is non-specific, the system will be *unstable* to fluctuations in the direction of the selected mode, and therefore will hallucinate spurious answers.

## 3  The Two Point System

A particularly simple case to consider has just two neurons (for the S system; two pairs of neurons for the EI system) and weights

$$\mathbf{J} = \begin{pmatrix} j_0 & j \\ j & j_0 \end{pmatrix} \qquad \mathbf{W} = \begin{pmatrix} w_0 & w \\ w & w_0 \end{pmatrix}$$

The idea is that each node coarsely models a group of neurons, and the interactions between neurons within a group ($j_o$ and $w_o$) are qualitatively different from interactions between neurons between groups ($j$ and $w$). The form of selective amplification here is that symmetric or ambiguous inputs $\mathbf{I}^a = I(1,1)$ should be suppressed compared with asymmetric inputs $\mathbf{I}^b = I(1,0)$ (and, equivalently, $I(0,1)$). In particular, given $\mathbf{I}^a$, the system should not spontaneously generate a response with $x_1$ significantly different from $x_2$. Define the fixed points to be $\bar{x}_1^a = \bar{x}_2^a > T$ under $\mathbf{I}^a$ and $\bar{x}_1^b > T > \bar{x}_2^b$ under $\mathbf{I}^b$, where $T$ is the threshold of the excitatory neurons. These relationships will be true across a wide range of input levels $I$. The ratio

$$R = \frac{d\bar{x}_1^b/dI}{d\bar{x}_1^a/dI} = \frac{1 + ((w_o + w) - (j_o + j))}{1 + (w_o - j_o)} = 1 + \frac{(w - j)}{1 + (w_o - j_o)} \tag{4}$$

of the average relative responses as the input level $I$ changes is a measure of how the system selectively amplifies the preferred or consistent inputs against ambiguous ones. This measure is appropriate only when the fluctuations of the system

The symmetry preserving network     The symmetry breaking network

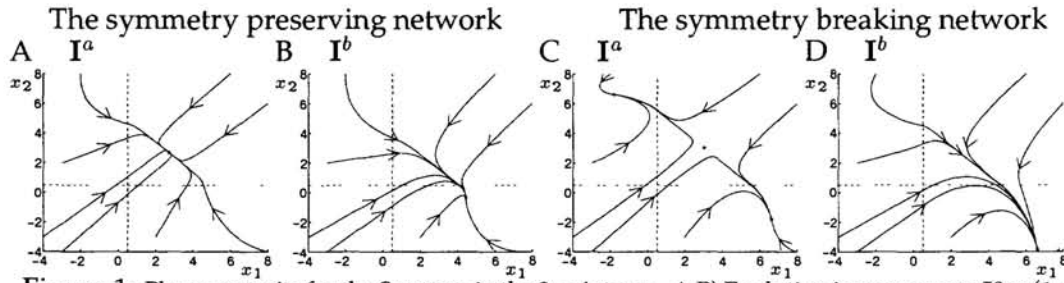

Figure 1: Phase portraits for the S system in the 2 point case. A;B) Evolution in response to $\mathbf{I}^a \propto (1,1)$ and $\mathbf{I}^b \propto (1,0)$ for parameters for which the response to $\mathbf{I}^a$ is stably symmetric. C;D) Evolution in response to $\mathbf{I}^a$ and $\mathbf{I}^b$ for parameters for which the symmetric response to $\mathbf{I}^a$ is unstable, inducing two extra equilibrium points. The dotted lines show the thresholds $T$ for $g(x)$.

from the fixed points $\bar{x}^a$ and $\bar{x}^b$ are well behaved. We will show that this requirement permits larger values of $R$ in the EI system than the S system, suggesting that the EI system can be a more powerful selective amplifier.

In the S system, the stabilities are governed by $\gamma^S = -(1 + w_o - j_o)$ for the single mode of deviation $x_1 - \bar{x}_1^b$ around fixed point $b$ and $\gamma_{\pm}^S = -(1 + (w_o \pm w) - (j_o \pm j))$ for the two modes of deviation $x_{\pm} \equiv (x_1 - \bar{x}_1^a) \pm (x_2 - \bar{x}_2^a)$ around fixed point $a$. Since we only consider cases when the input-output relationship $d\bar{x}/d\mathbf{I}$ of the fixed points is well defined, this means $\gamma^S < 0$ and $\gamma_+^S < 0$. However, for some interaction parameters, there are two extra (uneven) fixed points $\bar{x}_1^a \neq \bar{x}_2^a$ for (the even) input $I^a$. Dynamic systems theory dictates these two uneven fixed points will be stable and that they will appear when the '$-$' mode of the perturbation around the even fixed point $\bar{x}_1^a = \bar{x}_2^a$ is unstable. The system breaks symmetry in inputs, *ie* the motion trajectory diverges from the (unstable) even fixed point to one of the (stable) uneven ones. To avoid such cases, it is necessary that $\gamma_-^S < 0$. Combining this condition with equation 4 and $\gamma^S < 0$ leads to a upper bound on the amplification ratio $R^S < 2$. Figure 1 shows phase portraits and the equilibrium points of the S system under input $I^a$ and $I^b$ for the two different system parameter regions.

As we have described, the EI system has exactly the same fixed points as the S system, but they are more unstable. The stability around the symmetric fixed point under $\mathbf{I}^a$ is governed by $\gamma_{\pm}^{EI} = -1 + (j_o \pm j)/2 \pm \sqrt{(j_o \pm j)^2/4 - (w_o \pm w)}$, while that of the asymmetric fixed point under $\mathbf{I}^b$ or $\mathbf{I}^a$ by $\gamma^{EI} = -1 + j_o/2 \pm \sqrt{j_o^2/4 - w_o}$. Consequently, when there are three fixed points under $\mathbf{I}^a$, all of them can be unstable in the EI system, and the motion trajectory cannot converge to any of them. In this case, when both the '$+$' and '$-$' modes around the symmetric fixed point $\bar{x}_1^a = \bar{x}_2^a$ are unstable, the global dynamics constrains the motion trajectory to a limit cycle around the fixed points. If $x_1^a \approx x_2^a$ on this limit cycle, then the EI system will not break symmetry, even though the selective amplification ratio $R > 2$. Figure 2 demonstrates the performance of the EI system in this regime. Figure 2A;B show various aspects of the response to input $\mathbf{I}^a$ which should be comparatively suppressed. The system oscillates in such a way that $x_1$ and $x_2$ tend to be extremely similar (including being synchronised). Figure 2C;D show the same aspects of the response to $\mathbf{I}^b$, which should be amplified. Again the network oscillates, and, although $g(x_2)$ is not driven completely to 0 (it peaks at 15), it is very strongly dominated by $g(x_1)$, and further, the overall response is much stronger than in figure 2A;B.

The pertinent difference between the EI and S systems is that while the S system (when $h(y)$ is linear) can only roll down the energy landscape to a stable fixed

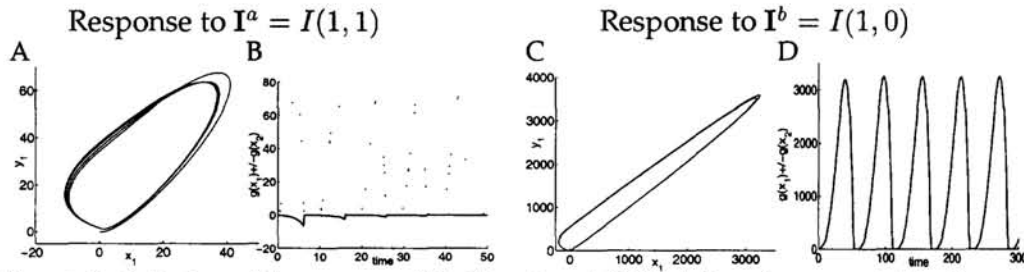

Figure 2: Projections of the response of the EI system. A;B) Evolution of response to $I^a$. A) $x_1$ vs $y_1$ and B) $g(x_1) - g(x_2)$ (solid); $g(x_1) + g(x_2)$ (dotted) across time show that the $x_1 = x_2$ mode dominates and the growth of $x_1 - x_2$ is strongly suppressed. C;D) Evolution of the response to $I^b$. Here, the response of $x_1$ always dominates that of $x_2$ over oscillations. The difference between $g(x_1) + g(x_2)$ and $g(x_1) - g(x_2)$ is too small to be evident on the figure. Note the difference in scales between A;B and C;D. Here $j_0 = 2.1; j = 0.4; w_0 = 1.11; w = 0.9$.

point and break the input symmetry, the EI system can resort to global limit cycles $x_1(t) \approx x_2(t)$ between unstable fixed points and maintain input symmetry. This is often (robustly over a large range of parameters) the case even when the '−' mode is locally *more* unstable (at the symmetric fixed point) than the '+' mode, because the '−' mode is much strongly suppressed when the motion trajectory enters the subthreshold region $x_1 < T$ and $x_2 < T$. As we can see in figure 2A;B, this acts to suppress any overall growth in the '−' mode. Since the asymmetric fixed point under $I^b$ is just as unstable as that under $I^a$, the EI system responds to asymmetric input $I^b$ also by a stable limit cycle around the asymmetric fixed point.

Since the response of the system in response to either pattern is oscillatory, there are various reasonable ways of evaluating the relative response ratio. Using the mean responses of the system during a cycle to define $\tilde{x}$, the selective amplification ratio in figure 2 is $R^{EI} = 97$, which is significantly higher than the $R^S = 2$ available from the S system. This is a simple existence proof of the superiority of the EI system for amplification, albeit at the expense of oscillations. In fact, in this two point case, it can be shown that any meaningful behavior of the S system (including symmetry breaking) can be qualitatively replicated in the EI system, but not vice-versa.

## 4   The Orientation System

Symmetric recurrent networks have recently been investigated in great depth for representing and calculating a wide variety of quantities, including orientation tuning. The idea behind the recurrent networks is that they should take noisy (and perhaps weakly tuned) input and selectively amplify the component that represents an orientation $\theta$ in the input, leaving a tuned pattern of excitation across the population that faithfully represents the underlying input. Based on the analysis above, we can expect that if an S network amplifies a tuned input enough, then it will break input symmetry given an untuned input and thus hallucinate a tuned response. However, an EI system, in the same oscillatory regime as for the two point system, can maintain untuned and suppressed response to untuned inputs.

We designed a particular EI system with a high selective amplification factor for tuned inputs $I(\theta)$. In this case, units $x_i, y_i$ have preferred orientations $\theta_i = (i - N/2)\pi/N$ for $i = 1 \dots n$. the connection matrices $J$ is Töplitz with Gaussian tuning, and, for simplicity, $[W]_{ij}$ does not depend on $i, j$. Figure 3B (and inset) shows the output of two units in the network in response to a tuned input, showing the nature of the oscillations and the way that selectivity builds up over the course of each period. Figure 3C shows the activities of all the units at three particular phases of the oscillation. Figure 3A shows how the mean activity of the most

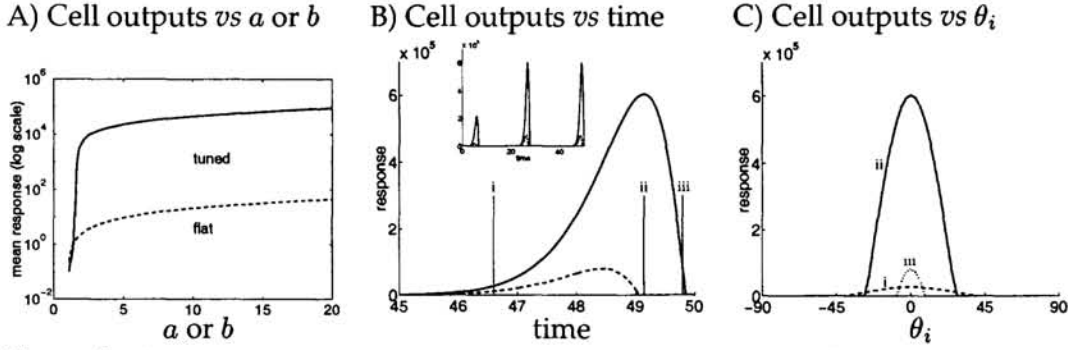

A) Cell outputs *vs a or b*    B) Cell outputs *vs* time    C) Cell outputs *vs* $\theta_i$

Figure 3: The Gaussian orientation network. A) Mean response of the $\theta_i = 0°$ unit in the network as a function of $a$ (untuned) or $b$ (tuned) with a log scale. B) Activity of the $\theta_i = 0°$ (solid) and $\theta_i = 30°$ (dashed) units in the network over the course of the positive part of an oscillation. Inset – activity of these units over all time. C) Activity of all the units at the three times shown as (i), (ii) and (iii) in (B) (i) (dashed) is in the rising phase of the oscillation; (ii) (solid) is at the peak; and (iii) (dotted) is during the falling phase. Here, the input is $I_i = a + be^{-\theta_i^2/2\sigma^2}$, with $\sigma = 13°$, and the Töplitz weights are $\mathbf{J}_{ij} = (3 + 21e^{-(\theta_i-\theta_j)^2/2\sigma'^2})/N$, with $\sigma' = 20°$ and $\mathbf{W}_{ij} = 23.5/N$.

activated unit scales with the levels of tuned and untuned input. The network amplifies the tuned inputs dramatically more – note the logarithmic scale. The S system breaks symmetry to the untuned input ($b = 0$) for these weights. If the weights are scaled uniformly by a factor of 0.22, then the S system is appropriately stable. However, the magnification ratio is 4.2 rather than something greater than 1000 in the EI system.

The orientation system can be understood to a large qualitative degree by looking at its two-point cousins. Many of the essential constraints on the system are determined by the behavior of the system when the mode with $x_i = x_j$ dominates, in which case the complex non-linearities induced by orientation tuning or cut off and its equivalents are irrelevant. Let $\tilde{J}(f)$ and $\tilde{W}(f)$ for (angular) frequency $f$ be the Fourier transforms of $J(i - j) \equiv [\mathbf{J}]_{ij}$ and $W(i - j) \equiv [\mathbf{W}]_{ij}$ and define $\lambda(f) = Re\{-1 + \tilde{J}(f)/2 + i\sqrt{(\tilde{W}(f) - \tilde{J}^2(f)/4)}\}$. Then, let $f^* > 0$ be the frequency such that $\lambda(f^*) \geq \lambda(f)$ for all $f > 0$. This is the non-translation-invariant mode that is most likely to cause instabilities for translation invariant behavior. A two point system that closely corresponds to the full system can be found by solving the simultaneous equations:

$$j_o + j = \tilde{J}(0) \quad w_o + w = \tilde{W}(0) \quad j_o - j = \tilde{J}(f^*) \quad w_o - w = \tilde{W}(f^*)$$

This design equates the $x_1 = x_2$ mode in the two point system with the $f = 0$ mode in the orientation system and the $x_1 = -x_2$ mode with the $f = f^*$ mode. For smooth $\mathbf{J}$ and $\mathbf{W}$, $f^*$ is often the smallest or one of the smallest non-zero spatial frequencies. It is easy to see that the two systems are exactly equivalent in the translation invariant mode $x_i = x_j$ under translation invariant input $I_i = I_j$ in both the linear and nonlinear regimes. The close correspondence between the two systems in other dynamic regimes is supported by simulation results.[16] Quantitatively, however, the amplification ratio differs between the two systems.

## 5  Conclusions

We have studied the dynamical behavior of networks with symmetrical and asymmetrical connections and have shown that the extra degrees of dynamical freedom

of the latter can be put to good computational use, *eg* global dynamic stability via local instability. Many applications of recurrent networks involve selective amplification – and the selective amplification factors for asymmetrical networks can greatly exceed those of symmetrical networks. We showed this in the case of orientation selectivity. However, it was originally inspired by a similar result in contour enhancement and texture segregation for which the activity of isolated oriented line elements should be enhanced if they form part of a smooth contour in the input and suppressed if they form part of an extended homogeneous texture. Further, the output should be homogeneous if the input is homogeneous (in the same way that the orientation network should not hallucinate orientations from untuned input). In this case, similar analysis[16] shows that stable contour enhancement is limited to just a factor of 3.0 for the S system (but not for the EI system), suggesting an explanation for the poor performance of a slew of S systems in the literature designed for this purpose. We used a very simple system with just two pairs of neurons to develop analytical intuitions which are powerful enough to guide our design of the more complex systems. We expect that the details of our model, with the exact pairing of excitatory and inhibitory cells and the threshold non-linearity, are not crucial for the results.

Inhibition in the cortex is, of course, substantially more complicated than we have suggested. In particular, inhibitory cells do have somewhat faster (though finite) time constants than excitatory cells, and are also not so subject to short term plasticity effects such as spike rate adaptation. Nevertheless, oscillations of various sorts can certainly occur, suggesting the relevance of the computational regime that we have studied.

# References

[1] Ben-Yishai, R, Bar-Or, RL & Sompolinsky, H (1995) *PNAS* **92**:3844-3848.
[2] Braun, J, Neibur, E, Schuster, HG & Koch, C (1994) *Society for Neuroscience Abstracts* **20**:1665.
[3] Carandini, M & Ringach, DL (1997) *Vision Research* **37**:3061-3071.
[4] Cohen, MA & Grossberg, S (1983) *IEEE Transactions on Systems, Man and Cybernetics* **13**:815-826.
[5] Eckhorn, R, *et al* (1988) *Biological Cybernetics* **60**:121-130.
[6] Ermentrout, GB & Cowan, JD (1979). *Journal of Mathematical Biology* **7**:265-280.
[7] Golomb, D, Wang, XJ & Rinzel, J (1996). *Journal of Neurophysiology* **75**:750-769.
[8] Gray, CM, Konig, P, Engel, AK & Singer, W (1989) *Nature* **338**:334-337.
[9] Grossberg, S (1988) *Neural Networks* **1**:17-61.
[10] Hopfield, JJ (1982) *PNAS* **79**:2554-2558.
[11] Hopfield, JJ (1984) *PNAS* **81**:3088-3092.
[12] Konig, P, Janosch, B & Schillen, TB (1992) *Neural Computation* **4**:666-681.
[13] Li, Z (1995) In JL van Hemmen *et al*, eds, *Models of Neural Networks*. Vol. 2. NY: Springer.
[14] Li, Z (1997) In KYM Wong, I King & DY Yeung, editors, *Theoretical Aspects of Neural Computation*. Hong Kong: Springer-Verlag.
[15] Li, Z (1998) *Neural Computation* **10**:903-940.
[16] Li, Z. and Dayan, P. (1999) to be published in *Network: Computations in Neural Systems*.
[17] Li, Z & Hopfield, JJ (1989). *Biological Cybernetics* **61**:379-392.
[18] Pouget, A, Zhang, KC, Deneve, S & Latham, PE (1998) *Neural Computation*, **10**373-401.
[19] Samsonovich A & McNaughton, BL (1997) *Journal of Neuroscience* **17**:5900-5920.
[20] Seung, HS (1996) *PNAS* **93**:13339-13344.
[21] Seung, HS *et al* (1998). *NIPS 10*.
[22] Sompolinsky, H, Golomb, D & Kleinfeld, D (1990) *PNAS* **87**:7200-7204.
[23] Stein, PSG, *et al* (1997) *Neurons, Networks, and Motor Behavior*. Cambridge, MA: MIT Press.
[24] Steriade, M, McCormick, DA & Sejnowski, TJ (1993). *Science* **262**:679-685.
[25] Suarez, H, Koch, C & Douglas, R (1995) *Journal of Neuroscience* **15**:6700-6719.
[26] von der Malsburg, C (1988) *Neural Networks* **1**:141-148.
[27] Zhang, K (1996) *Journal of Neuroscience* **16**:2112-2126.
